# A Hippocampal Model of Recognition Memory

**Randall C. O'Reilly**
Department of Psychology
University of Colorado at Boulder
Campus Box 345
Boulder, CO 80309-0345
oreilly@psych.colorado.edu

**Kenneth A. Norman**
Department of Psychology
Harvard University
33 Kirkland Street
Cambridge, MA 02138
norman@wjh.harvard.edu

**James L. McClelland**
Department of Psychology and
Center for the Neural Basis of Cognition
Carnegie Mellon University
Pittsburgh, PA 15213
jlm@cnbc.cmu.edu

## Abstract

A rich body of data exists showing that recollection of specific information makes an important contribution to recognition memory, which is distinct from the contribution of familiarity, and is not adequately captured by existing unitary memory models. Furthermore, neuropsychological evidence indicates that recollection is subserved by the hippocampus. We present a model, based largely on known features of hippocampal anatomy and physiology, that accounts for the following key characteristics of recollection: 1) false recollection is rare (i.e., participants rarely claim to recollect having studied nonstudied items), and 2) increasing interference leads to less recollection but apparently does not compromise the *quality* of recollection (i.e., the extent to which recollected information veridically reflects events that occurred at study).

## 1 Introduction

For nearly 50 years, memory researchers have known that our ability to remember specific past episodes depends critically on the hippocampus. In this paper, we describe our initial attempt to use a mechanistically explicit model of hippocampal function to explain a wide range of human memory data.

Our understanding of hippocampal function from a computational and biological perspec-

tive is based on our prior work (McClelland, McNaughton, & O'Reilly, 1995; O'Reilly & McClelland, 1994). At the broadest level, we think that the hippocampus exists in part to provide a memory system which can learn arbitrary information rapidly without suffering undue amounts of interference. This memory system sits on top of, and works in conjunction with, the neocortex, which learns slowly over many experiences, producing integrative representations of the relevant statistical features of the environment. The hippocampus accomplishes rapid, relatively interference-free learning by using relatively non-overlapping (*pattern separated*) representations. Pattern separation occurs as a result of 1) the *sparseness* of hippocampal representations (relative to cortical representations), and 2) the fact that hippocampal units are sensitive to *conjunctions* of cortical features — given two cortical patterns with 50% feature overlap, the probability that a particular conjunction of features will be present in both patterns is much less than 50%.

We propose that the hippocampus produces a relatively high-threshold, high-quality recollective response to test items. The response is "high-threshold" in the sense that studied items sometimes trigger rich recollection (defined as "retrieval of most or all of the test probe's features from memory") but lures never trigger rich recollection. The response is "high-quality" in the sense that, most of the time, the recollection signal consists of part or all of a single studied pattern, as opposed to a blend of studied patterns. The high-threshold, high-quality nature of recollection can be explained in terms of the conjunctivity of hippocampal representations: Insofar as recollection is a function of whether the features of the test probe were encountered *together* at study, lures (which contain many novel feature conjunctions, even if their constituent features are familiar) are unlikely to trigger rich recollection; also, insofar as the hippocampus stores feature conjunctions (as opposed to individual features), features which appeared together at study are likely to appear together at test. Importantly, in accordance with dual-process accounts of recognition memory (Yonelinas, 1994; Jacoby, Yonelinas, & Jennings, 1996), we believe that hippocampally-driven recollection is not the sole contributor to recognition memory performance. Rather, extensive evidence exists that recollection is complemented by a "fallback" familiarity signal which participants consult when rich recollection does not occur. The familiarity signal is mediated by as-yet unspecified areas (likely including the parahippocampal temporal cortex: Aggleton & Shaw, 1996; Miller & Desimone, 1994).

Our account differs substantially from most other computational and mathematical models of recognition memory. Most of these models compute the "global match" between the test probe and stored memories (e.g., Hintzman, 1988; Gillund & Shiffrin, 1984); recollection in these models involves computing a similarity-weighted average of stored memory patterns. In other memory models, recollection of an item depends critically on the extent to which the components of the item's representation were linked with that of the study context (e.g., Chappell & Humphreys, 1994). Critically, recollection in all of these models lacks the high-threshold, high-quality character of recollection in our model. This is most evident when we consider the effects of manipulations which increase interference (e.g., increasing the length of the study list, or increasing inter-item similarity). As interference increases, global matching models predict increasingly "blurry" recollection (reflecting the contribution of more items to the composite output vector), while the other models predict that false recollection of lures will increase. In contrast, our model predicts that increasing interference should lead to decreased correct recollection of studied test probes, but there should be no concomitant increase in "erroneous" types of recollection (i.e., recollection of details which mismatch studied test probes, or rich recollection of lures). This prediction is consistent with the recent finding that correct recollection of studied items decreases with increasing list length (Yonelinas, 1994). Lastly, although extant data certainly do not contradict the claim that the veridicality of recollection is robust to interference, we acknowledge that additional, focused experimentation is needed to definitively resolve this issue.

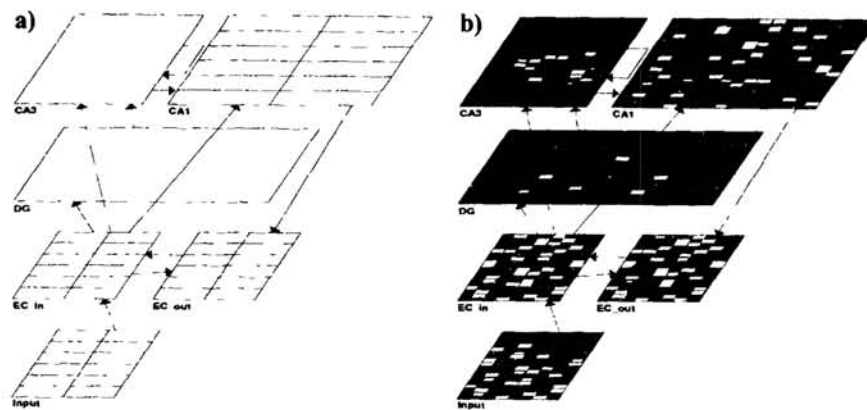

Figure 1: The model. **a)** Shows the areas and connectivity, and the corresponding columns within the Input, EC, and CA1 (see text). **b)** Shows an example activity pattern. Note the sparse activity in the DG and CA3, and intermediate sparseness of the CA1.

## 2   Architecture and Overall Behavior

Figure 1 shows a diagram of our model, which contains the basic anatomical regions of the hippocampal formation, as well as the entorhinal cortex (EC), which serves as the primary cortical input/output pathway for the hippocampus. The model as described below instantiates a series of hypotheses about the structure and function of the hippocampus and associated cortical areas, which are based on anatomical and physiological data and other models as described in O'Reilly and McClelland (1994) and McClelland et al. (1995), but not elaborated upon significantly here.

The *Input* layer activity pattern represents the state of the EC resulting from the presentation of a given item. We assume that the hippocampus stores and retrieves memories by way of reduced representations in the EC, which have a correspondence with more elaborated representations in other areas of cortex that is developed via long-term cortical learning. We further assume that there is a rough topology to the organization of EC, with different cortical areas and/or sub-areas represented by different *slots*, which can be thought of as representing different feature dimensions of the input (e.g., color, font, semantic features, etc.). Our EC has 36 slots with four units per slot; one unit per slot was active (with each unit representing a particular "feature value"). Input patterns were constructed from prototypes by randomly selecting different feature values for a random subset of slots. There are two functionally distinct layers of the EC, one which receives input from cortical areas and projects into the hippocampus (superficial or $EC_{in}$), and another which receives projections from the CA1 and projects back out to the cortex (deep or $EC_{out}$). While the representations in these layers are probably different in their details, we assume that they are functionally equivalent, and use the same representations across both for convenience. $EC_{in}$ projects to three areas of the hippocampus: the dentate gyrus (DG), area CA3, and area CA1. The storage of the input pattern occurs through weight changes in the feedforward and recurrent projections into the CA3, and the CA3 to CA1 connections. The CA3 and CA1 contain the two primary representations of the input pattern, while the DG plays an important but secondary role as a pattern-separation enhancer for the CA3.

The CA3 provides the primary sparse, pattern-separated, conjunctive representation described above. This is achieved by random, partial connectivity between the EC and CA3, and a high threshold for activation (i.e., sparseness), such that the few units which are activated in the CA3 (5% in our model) are those which have the most inputs from active EC units. The odds of a unit having such a high proportion of inputs from even two relatively similar EC patterns is low, resulting in pattern separation (see O'Reilly & McClelland,

1994 for a much more detailed and precise treatment of this issue, and the role of the DG in facilitating pattern separation). While these CA3 representations are useful for allowing rapid learning without undue interference, the pattern-separation process eliminates any systematic relationship between the CA3 pattern and the original EC pattern that gave rise to it. Thus, there must be some means of translating the CA3 pattern back into the language of the EC. The simple solution of directly associating the CA3 pattern with the corresponding EC pattern is problematic due to the interference caused by the relatively high activity levels in the EC (around 15%, and 25% in our model). For this reason, we think that the translation is formed via the CA1, which (as a result of long-term learning) is capable of expanding EC representations into sparser patterns that are more easily linked to CA3, and then mapping these sparser patterns back onto the EC.

Our CA1 has separate representations of small combinations of slots (labeled *columns*); columns can be arbitrarily combined to reproduce any valid EC representation. Thus, representations in CA1 are intermediate between the fully conjunctive CA3, and the fully combinatorial EC. This is achieved in our model by training a single CA1 column of 32 units with slightly less than 10% activity levels to be able to reproduce any combination of patterns over 3 $EC_{in}$ slots (64 different combinations) in a corresponding set of 3 $EC_{out}$ slots. The resulting weights are replicated across columns covering the entire EC (see Figure 1a). The cost of this scheme is that more CA1 units are required (32 *vs* 12 per column in the EC), which is nonetheless consistent with the relatively greater expansion of this area relative to other hippocampal areas as a function of cortical size.

After learning, our model recollects studied items by simply reactivating the original CA3, CA1 and $EC_{out}$ patterns via facilitated weights. With partial or noisy input patterns (and with interference), these weights and two forms of recurrence (the "short loop" within CA3, and the "big loop" out to the EC and back through the entire hippocampus) allow the hippocampus to bootstrap its way into recalling the complete original pattern (*pattern completion*). If the EC input pattern corresponds to a nonstudied pattern, then the weights will not have been facilitated for this particular activity pattern, and the CA1 will not be strongly driven by the CA3. Even if the $EC_{in}$ activity pattern corresponds to two components that were previously studied, but not together (see below), the conjunctive nature of the CA3 representations will minimize the extent to which recall occurs.

Recollection is operationalized as successful recall of the test probe. This raises the basic problem that the system needs to be able to distinguish between the $EC_{out}$ activation due to the item input on $EC_{in}$ (either directly or via the CA1), and that which is due to activation coming from recall in the CA3-CA1 pathway. One solution to this problem, which is suggested by autocorrelation histograms during reversible CA3 lesions (Mizumori et al., 1989), is that the CA3 and CA1 are 180° out of phase with respect to the theta rhythm. Thus, when the CA3 drives the CA1, it does so at a point when the CA1 units would otherwise be silent, providing a means for distinguishing between EC and CA3 driven CA1 activation. We approximate something like this mechanism by simply turning off the $EC_{in}$ inputs to CA1 during testing. We assess the quality of hippocampal recall by comparing the resulting $EC_{out}$ pattern with the $EC_{in}$ cue. The number of active units that match between $EC_{in}$ and $EC_{out}$ (labeled $C$) indicates how much of the test probe was recollected. The number of units that are active in $EC_{out}$ but not in $EC_{in}$ (labeled $E$) indicates the extent to which the model recollected an item other than the test probe.

## 3   Activation and Learning Dynamics

Our model is implemented using the Leabra framework, which provides a robust mechanism for producing controlled levels of sparse activation in the presence of recurrent activa-

tion dynamics, and a simple, effective Hebbian learning rule (O'Reilly, 1996)[1]. The activation function is a simple thresholded single-compartment neuron model with continuous-valued spike rate output. Membrane potential is updated by $\frac{dV_m(t)}{dt} = \tau \sum_c g_c(t)\overline{g_c}(E_c - V_m(t))$, with 3 channels ($c$) corresponding to: $e$ excitatory input; $l$ leak current; and $i$ inhibitory input. Activation communicated to other cells is a simple thresholded function of the membrane potential: $y_j(t) = 1/\left(1 + \frac{1}{\gamma[V_m(t)-\Theta]_+}\right)$. As in the hippocampus (and cortex), all principal weights (synaptic efficacies) are excitatory, while the local-circuit inhibition controls positive feedback loops (i.e., preventing epileptiform activity) and produces sparse representations. Leabra assumes that the inhibitory feedback has an approximate set-point (i.e., strong activity creates compensatorially stronger inhibition, and vice-versa), resulting in roughly constant overall activity levels, with a firm upper bound. Inhibitory current is given by $g_i = g_{k+1}^\Theta + q(g_k^\Theta - g_{k+1}^\Theta)$, where $0 < q < 1$ is typically .25, and $g^\Theta = \frac{\sum_{c \neq i} g_c \overline{g_c}(E_c - \Theta)}{\Theta - E_i}$ for the units with the $k$ th and $k + 1$ th highest excitatory inputs. A simple, appropriately normalized Hebbian rule is used in Leabra: $\Delta w_{ij} = x_i y_j - y_j w_{ij}$, which can be seen as computing the expected value of the sending unit's activity conditional on the receiver's activity (if treated like a binary variable active with probability $y_j$): $w_{ij} \approx \langle x_i | y_j \rangle_p$. This is essentially the same rule used in standard competitive learning or mixtures-of-Gaussians.

## 4  Interference and List-Length, Item Similarity

Here, we demonstrate that the hippocampal recollection system degrades with increasing interference in a way that preserves its essential high-threshold, high-quality nature. Figure 2 shows the effects of list length and item similarity on our $C$ and $E$ measures. Only studied items appear in the high $C$, low $E$ corner representing rich recollection. As length and similarity increase, interference results in decreased $C$ for studied items (without increased $E$), but critically there is no change in responding to new items. Interference in our model arises from the reduced but nevertheless extant overlap between representations in the hippocampal system as a function of item similarity and number of items stored. To the extent that increasing numbers of individual CA3 units are linked to multiple contradictory CA1 representations, their contribution is reduced, and eventually recollection fails. As for the frequently obtained finding that decreased recollection of studied items is accompanied by an increase in overall false alarms, we think this results from subjects being forced to rely more on the (less reliable) fallback familiarity mechanism.

## 5  Conjunctivity and Associative Recognition

Now, we consider what happens when lures are constructed by recombining elements of studied patterns (e.g., study "window-reason" and "car-oyster", and test with "window-oyster"). One recent study found that participants are much more likely to claim to recollect studied pairs than re-paired lures (Yonelinas, 1997). Furthermore, data from this study is consistent with the idea that re-paired lures sometimes trigger recollection of the studied word pairs that were re-combined to generate the lure; when this happens (assuming that each word occurred in only one pair), the participant can confidently reject the lure. Our simulation data is consistent with these findings: For studied word pairs, the model (richly) recollected both pair components 86% of the time. As for re-paired lures, both pair components were never recalled together, but 16% of the time the model recollected one of the pair components, along with the component that it was paired with at study. The

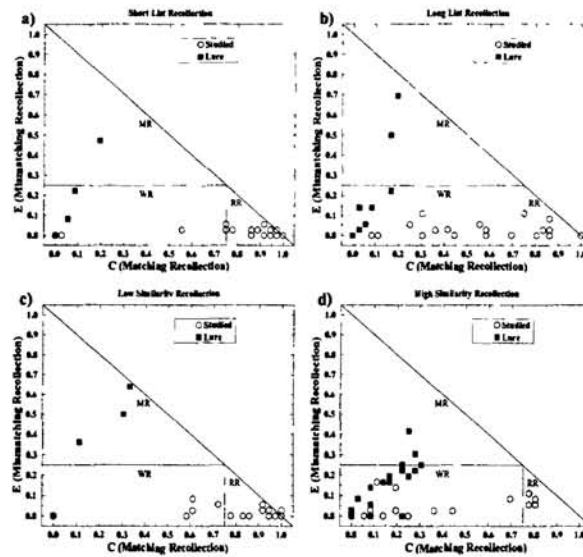

Figure 2: Effects of list length and similarity on recollection performance. Responses can be categorized according to the thresholds shown, producing three regions: rich recollection (*RR*), weak recollection (*WR*), and misrecollection (*MR*). Increasing list length and similarity lead to less rich recollection of studied items (without increasing misrecollection for these items), and do not significantly affect the model's responding to lures.

model responded in a similar fashion to pairs consisting of one studied word and a new word (never recollecting both pair components together, but recollecting the old item and the item it was paired with at study 13% of the time). Word pairs consisting of two new items failed to trigger recollection of even a single pair component. Similar findings were obtained in our simulation of the (Hintzman, Curran, & Oppy, 1992) experiment involving recombinations of word and plurality cues.

## 6  Discussion

While the results presented above have dealt with the presentation of complete probe stimuli for recognition memory tests, our model is obviously capable of explaining cued recall and related phenomena such as source or context memory by virtue of its pattern completion abilities. There are a number of interesting issues that this raises. For example, we predict that successful item recollection will be highly correlated with the ability to recall additional information from the study episode, since both rely on the same underlying memory. Further, to the extent that elderly adults form less distinct encodings of stimuli (Rabinowitz & Ackerman, 1982), this explains both their impaired recollection on recognition tests (Parkin & Walter, 1992) and their impaired memory for contextual ("source") details (Schacter et al., 1991).

In summary, existing mathematical models of recognition memory are most likely incorrect in assuming that recognition is performed with one memory system. Global matching models may provide a good account of familiarity-based recognition, but they fail to account for the contributions of recollection to recognition, as discussed above. For example, global matching models predict that lures which are similar to studied items will always trigger a stronger signal than dissimilar lures; as such, these models can not account for the fact that sometimes subjects can reject similar lures with high levels of confidence (due, in our model, to recollection of a similar studied item; Brainerd, Reyna, & Kneer, 1995; Hintzman et al., 1992). Further, global matching models confound the signal for the extent to which individual components of the test probe were present at all during study, and signal for the

extent to which they occurred together. We believe that these signals may be separable, with recollection (implemented by the hippocampus) showing sensitivity to conjunctions of features, but not the occurrence of individual features, and familiarity (implemented by cortical regions) showing sensitivity to component occurrence but not co-occurence. This division of labor is consistent with recent data showing that familiarity does not discriminate well between studied item pairs and lures constructed by conjoining items from two different studied pairs (so long as the pairings are truly novel) (Yonelinas, 1997), and with the point, set forth by (McClelland et al., 1995), that catastrophic interference would occur if rapid learning (required to learn feature co-occurrences) took place in the neocortical structures which generate the familiarity signal.

# 7 References

Aggleton, J. P., & Shaw, C. (1996). Amnesia and recognition memory: a re-analysis of psychometric data. *Neuropsychologia, 34*, 51.

Brainerd, C. J., Reyna, V. F., & Kneer, R. (1995). False-recognition reversal: When similarity is distinctive. *Journal of Memory and Language, 34*, 157–185.

Chappell, M., & Humphreys, M. S. (1994). An auto-associative neural network for sparse representations: Analysis and application to models of recognition and cued recall. *Psychological Review, 101*, 103–128.

Gillund, G., & Shiffrin, R. M. (1984). A retrieval model for both recognition and recall. *Psychological Review, 91*, 1–67.

Hintzman, D. L. (1988). Judgments of frequency and recognition memory in a multiple-trace memory model. *Psychological Review, 95*, 528–551.

Hintzman, D. L., Curran, T., & Oppy, B. (1992). Effects of similiarity and repetition on memory: Registration without learning. *Journal of Experimental Psychology: Learning, Memory, and Cognition, 18*, 667–680.

Jacoby, L. L., Yonelinas, A. P., & Jennings, J. M. (1996). The relation between conscious and unconscious (automatic) influences: A declaration of independence. In J. D. Cohen, & J. W. Schooler (Eds.), *Scientific approaches to the question of consciousness* (pp. 13–47). Hillsdale, NJ: Lawrence Erbaum Associates.

McClelland, J. L., McNaughton, B. L., & O'Reilly, R. C. (1995). Why there are complementary learning systems in the hippocampus and neocortex: Insights from the successes and failures of connectionst models of learning and memory. *Psychological Review, 102*, 419–457.

Miller, E. K., & Desimone, R. (1994). Parallel neuronal mechanisms for short-term memory. *Science, 263*, 520–522.

Mizumori, S. J. Y., McNaughton, B. L., Barnes, C. A., & Fox, K. B. (1989). Preserved spatial coding in hippocampal CA1 pyramidal cells during reversible suppression of CA3c output: Evidence for pattern completion in hippocampus. *Journal of Neuroscience, 9*(11), 3915–3928.

O'Reilly, R. C. (1996). *The leabra model of neural interactions and learning in the neocortex*. PhD thesis, Carnegie Mellon University, Pittsburgh, PA, USA.

O'Reilly, R. C., & McClelland, J. L. (1994). Hippocampal conjunctive encoding, storage, and recall: Avoiding a tradeoff. *Hippocampus, 4*(6), 661–682.

Parkin, A. J., & Walter, B. M. (1992). Recollective experience, normal aging, and frontal dysfunction. *Psychology and Aging, 7*, 290–298.

Rabinowitz, J. C., & Ackerman, B. P. (1982). General encoding of episodic events by elderly adults. In F. I. M. C. S. Trehub (Ed.), *Aging and cognitive processes*. Plenum Publishing Corporation.

Schacter, D. L., Kaszniak, A. W., Kihlstrom, J. F., & Valdiserri, M. (1991). The relation between source memory and aging. *Psychology and Aging, 6*, 559–568.

Yonelinas, A. P. (1994). Receiver-operating characteristics in recognition memory: Evidence for a dual-process model. *Journal of Experimental Psychology: Learning, Memory, and Cognition, 20*, 1341–1354.

Yonelinas, A. P. (1997). Recognition memory ROCs for item and associative information: The contribution of recollection and familiarity. *Memory and Cognition, 25*, 747–763.

## Footnotes

[1]Note that the version of Leabra described here is an update to the cited version, which is currently being prepared for publication.
